# Products of Gaussians

**Christopher K. I. Williams**
Division of Informatics
University of Edinburgh
Edinburgh EH1 2QL, UK
*c.k.i.williams@ed.ac.uk*
*http://anc.ed.ac.uk*

**Felix V. Agakov**
System Engineering Research Group
Chair of Manufacturing Technology
Universität Erlangen-Nürnberg
91058 Erlangen, Germany
*F.Agakov@lft.uni-erlangen.de*

**Stephen N. Felderhof**
Division of Informatics
University of Edinburgh
Edinburgh EH1 2QL, UK
*stephenf@dai.ed.ac.uk*

## Abstract

Recently Hinton (1999) has introduced the Products of Experts (PoE) model in which several individual probabilistic models for data are combined to provide an overall model of the data. Below we consider PoE models in which each expert is a Gaussian. Although the product of Gaussians is also a Gaussian, if each Gaussian has a simple structure the product can have a richer structure. We examine (1) Products of Gaussian pancakes which give rise to probabilistic Minor Components Analysis, (2) products of 1-factor PPCA models and (3) a products of experts construction for an AR(1) process.

Recently Hinton (1999) has introduced the Products of Experts (PoE) model in which several individual probabilistic models for data are combined to provide an overall model of the data. In this paper we consider PoE models in which each expert is a Gaussian. It is easy to see that in this case the product model will also be Gaussian. However, if each Gaussian has a simple structure, the product can have a richer structure. Using Gaussian experts is attractive as it permits a thorough analysis of the product architecture, which can be difficult with other models, e.g. models defined over discrete random variables.

Below we examine three cases of the products of Gaussians construction: (1) Products of Gaussian pancakes (PoGP) which give rise to probabilistic Minor Components Analysis (MCA), providing a complementary result to probabilistic Principal Components Analysis (PPCA) obtained by Tipping and Bishop (1999); (2) Products of 1-factor PPCA models; (3) A products of experts construction for an AR(1) process.

**Products of Gaussians**

If each expert is a Gaussian $p_i(\mathsf{x}|\Theta_i) \sim N(\mu_i, \mathsf{C}_i)$, the resulting distribution of the product of $m$ Gaussians may be expressed as

$$p(\mathsf{x}|\Theta) \propto \exp\left\{ -\frac{1}{2} \sum_{i=1}^{m} (\mathsf{x} - \mu_i)^T \mathsf{C}_i^{-1} (\mathsf{x} - \mu_i) \right\}.$$

By completing the square in the exponent it may be easily shown that $p(\mathsf{x}|\Theta) \sim N(\mu_\Sigma, \mathsf{C}_\Sigma)$, where $\mathsf{C}_\Sigma^{-1} = \sum_{i=1}^{m} \mathsf{C}_i^{-1}$. To simplify the following derivations we will assume that $p_i(\mathsf{x}|\Theta_i) \sim N(0, \mathsf{C}_i)$ and thus that $p(\mathsf{x}|\Theta) \sim N(0, \mathsf{C}_\Sigma)$. $\mu_\Sigma \neq 0$ can be obtained by translation of the coordinate system.

# 1 Products of Gaussian Pancakes

A Gaussian "pancake" (GP) is a $d$-dimensional Gaussian, contracted in one dimension and elongated in the other $d-1$ dimensions. In this section we show that the maximum likelihood solution for a product of Gaussian pancakes (PoGP) yields a probabilistic formulation of Minor Components Analysis (MCA).

## 1.1 Covariance Structure of a GP Expert

Consider a $d$-dimensional Gaussian whose probability contours are contracted in the direction $\hat{\mathsf{w}}$ and equally elongated in mutually orthogonal directions $\mathsf{v}_1, \ldots, \mathsf{v}_{d-1}$. We call this a Gaussian pancake or GP. Its inverse covariance may be written as

$$\mathsf{C}^{-1} = \sum_{i=1}^{d-1} \mathsf{v}_i \mathsf{v}_i^T \beta_0 + \hat{\mathsf{w}}\hat{\mathsf{w}}^T \beta_{\hat{w}}, \tag{1}$$

where $\mathsf{v}_1, \ldots, \mathsf{v}_{d-1}, \hat{\mathsf{w}}$ form a $d \times d$ matrix of normalized eigenvectors of the covariance $\mathsf{C}$. $\beta_0 = \sigma_0^{-2}$, $\beta_{\hat{w}} = \sigma_{\hat{w}}^{-2}$ define inverse variances in the directions of elongation and contraction respectively, so that $\sigma_0^2 \geq \sigma_{\hat{w}}^2$. Expression (1) can be re-written in a more compact form as

$$\mathsf{C}^{-1} = \beta_0 \mathsf{I}_d + (\beta_{\hat{w}} - \beta_0)\hat{\mathsf{w}}\hat{\mathsf{w}}^T = \beta_0 \mathsf{I}_d + \mathsf{w}\mathsf{w}^T, \tag{2}$$

where $\mathsf{w} = \hat{\mathsf{w}}\sqrt{\beta_{\hat{w}} - \beta_0}$ and $\mathsf{I}_d \subset \mathbb{R}^{d \times d}$ is the identity matrix. Notice that according to the constraint considerations $\beta_0 < \beta_{\hat{w}}$, and all elements of $\mathsf{w}$ are real-valued.

Note the similarity of (2) with expression for the covariance of the data of a 1-factor probabilistic principal component analysis model $\mathsf{C} = \sigma^2 \mathsf{I}_d + \mathsf{w}\mathsf{w}^T$ (Tipping and Bishop, 1999), where $\sigma^2$ is the variance of the factor-independent spherical Gaussian noise. The only difference is that it is the *inverse* covariance matrix for the constrained Gaussian model rather than the covariance matrix which has the structure of a rank-1 update to a multiple of $\mathsf{I}_d$.

## 1.2 Covariance of the PoGP Model

We now consider a product of $m$ GP experts, each of which is contracted in a single dimension. We will refer to the model as a $(1, m)$ PoGP, where 1 represents the number of directions of contraction of each expert. We also assume that all experts have identical means.

From (1), the inverse covariance of the the resulting $(1, m)$ PoGP model can be expressed as

$$C_\Sigma^{-1} = \sum_{i=1}^{m} C_i^{-1} = \beta_\Sigma I_d + WW^T \qquad (3)$$

where columns of $W \subset \mathbb{R}^{d \times m}$ correspond to weight vectors of the $m$ PoGP experts, and $\beta_\Sigma = \sum_{i=1}^{m} \beta_0^{(i)} > 0$.

## 1.3 Maximum-Likelihood Solution for PoGP

Comparing (3) with $m$-factor PPCA we can make a conjecture that in contrast with the PPCA model where ML weights correspond to principal components of the data covariance (Tipping and Bishop, 1999), weights $W$ of the PoGP model define projection onto $m$ *minor* eigenvectors of the sample covariance in the visible $d$-dimensional space, while the distortion term $\beta_\Sigma I_d$ explains larger variations[1]. This is indeed the case.

In Williams and Agakov (2001) it is shown that stationarity of the log-likelihood with respect to the weight matrix $W$ and the noise parameter $\beta_\Sigma$ results in three classes of solutions for the experts' weight matrix, namely

$$\begin{aligned}
W &= 0; \\
S &= C_\Sigma; \\
SW &= C_\Sigma W, \quad W \neq 0, \quad S \neq C_\Sigma,
\end{aligned} \qquad (4)$$

where $S$ is the covariance matrix of the data (with an assumed mean of zero). The first two conditions in (4) are the same as in Tipping and Bishop (1999), but for PPCA the third condition is replaced by $C^{-1}W = S^{-1}W$ (assuming that $S^{-1}$ exists). In Appendix A and Williams and Agakov (2001) it is shown that the maximum likelihood solution for $W_{ML}$ is given by:

$$W_{ML} = U(\Lambda^{-1} - \beta_\Sigma^{ML} I_m)^{1/2} R^T, \qquad \beta_\Sigma^{ML} = \frac{d - m}{\sum_{i=m+1}^{d} \lambda_i}, \qquad (5)$$

where $R \subset \mathbb{R}^{m \times m}$ is an arbitrary rotation matrix, $\Lambda$ is a $m \times m$ matrix containing the $m$ smallest eigenvalues of $S$ and $U = [u_1, \dots, u_m] \subset \mathbb{R}^{d \times m}$ is a matrix of the corresponding eigenvectors of $S$. Thus, the maximum likelihood solution for the weights of the $(1, m)$ PoGP model corresponds to $m$ scaled and rotated minor eigenvectors of the sample covariance $S$ and leads to a probabilistic model of minor component analysis. As in the PPCA model, the number of experts $m$ is assumed to be lower than the dimension of the data space $d$.

The correctness of this derivation has been confirmed experimentally by using a scaled conjugate gradient search to optimize the log likelihood as a function of $W$ and $\beta_\Sigma$.

## 1.4 Discussion of PoGP model

An intuitive interpretation of the PoGP model is as follows: Each Gaussian pancake imposes an approximate linear constraint in x space. Such a linear constraint is that x should lie close to a particular hyperplane. The conjunction of these constraints is given by the product of the Gaussian pancakes. If $m \ll d$ it will make sense to

define the resulting Gaussian distribution in terms of the constraints. However, if there are many constraints ($m > d/2$) then it can be more efficient to describe the directions of large variability using a PPCA model, rather than the directions of small variability using a PoGP model. This issue is discussed by Xu et al. (1991) in what they call the "Dual Subspace Pattern Recognition Method" where both PCA and MCA models are used (although their work does not use explicit probabilistic models such as PPCA and PoGP).

MCA can be used, for example, for signal extraction in digital signal processing (Oja, 1992), dimensionality reduction, and data visualization. Extraction of the minor component is also used in the Pisarenko Harmonic Decomposition method for detecting sinusoids in white noise (see, e.g. Proakis and Manolakis (1992), p. 911). Formulating minor component analysis as a probabilistic model simplifies comparison of the technique with other dimensionality reduction procedures, permits extending MCA to a mixture of MCA models (which will be modeled as a mixture of products of Gaussian pancakes), permits using PoGP in classification tasks (if each PoGP model defines a class-conditional density), and leads to a number of other advantages over non-probabilistic MCA models (see the discussion of advantages of PPCA over PCA in Tipping and Bishop (1999)).

## 2 Products of PPCA

In this section we analyze a product of $m$ 1-factor PPCA models, and compare it to a $m$-factor PPCA model.

### 2.1 1-factor PPCA model

Consider a 1-factor PPCA model, having a latent variable $s_i$ and visible variables $\mathsf{x}$. The joint distribution is given by $P(s_i, \mathsf{x}) = P(s_i)P(\mathsf{x}|s_i)$. We set $P(s_i) \sim N(0,1)$ and $P(\mathsf{x}|s_i) \sim N(\mathsf{w}_i s_i, \sigma^2)$. Integrating out $s_i$ we find that $P_i(\mathsf{x}) \sim N(0, \mathsf{C}_i)$ where $\mathsf{C}_i = \mathsf{w}_i \mathsf{w}_i^T + \sigma^2 \mathsf{I}_d$ and

$$\mathsf{C}_i^{-1} = \beta \mathsf{I}_d - \frac{\beta^2 \mathsf{w}_i \mathsf{w}_i^T}{1 + \beta \mathsf{w}_i^T \mathsf{w}_i} = \beta \mathsf{I}_d - \beta \gamma_i \mathsf{w}_i \mathsf{w}_i^T, \tag{6}$$

where $\beta = \sigma^{-2}$ and $\gamma_i = \beta/(1 + \beta \|\mathsf{w}_i\|^2)$. $\beta$ and $\gamma_i$ are the inverse variances in the directions of contraction and elongation respectively.

The joint distribution of $s_i$ and $\mathsf{x}$ is given by

$$P(\mathsf{x}, s_i) \quad \propto \quad \exp -\frac{1}{2} \left[ s_i^2 + \beta(\mathsf{x} - \mathsf{w}_i s_i)^T (\mathsf{x} - \mathsf{w}_i s_i) \right] \tag{7}$$

$$= \quad \exp -\frac{\beta}{2} \left[ \frac{s_i^2}{\gamma_i} - 2\mathsf{x}^T \mathsf{w}_i s_i + \mathsf{x}^T \mathsf{x} \right]. \tag{8}$$

Tipping and Bishop (1999) showed that the general $m$-factor PPCA model ($m$-PPCA) has covariance $\mathsf{C} = \sigma^2 \mathsf{I}_d + \mathsf{WW}^T$, where $\mathsf{W}$ is the $d \times m$ matrix of factor loadings. When fitting this model to data, the maximum likelihood solution is to choose $\mathsf{W}$ proportional to the principal components of the data covariance matrix.

## 2.2 Products of 1-factor PPCA models

We now consider the product of $m$ 1-factor PPCA models, which we denote a $(1, m)$-PoPPCA model. The joint distribution over $\mathsf{s} = (s_1, \ldots, s_m)^T$ and $\mathsf{x}$ is

$$P(\mathsf{x}, \mathsf{s}) \propto \exp{-\frac{\beta}{2} \sum_{i=1}^{m} \left[ \frac{s_i^2}{\gamma_i} - 2\mathsf{x}^T \mathsf{w}_i s_i + \mathsf{x}^T \mathsf{x} \right]}. \tag{9}$$

Let $\mathsf{z}^T \stackrel{\text{def}}{=} (\mathsf{x}^T, \mathsf{s}^T)$. Thus we see that the distribution of $\mathsf{z}$ is Gaussian with inverse covariance matrix $\beta \mathsf{M}$, where

$$\mathsf{M} = \begin{pmatrix} m\mathsf{I}_d & -\mathsf{W} \\ -\mathsf{W}^T & \Gamma^{-1} \end{pmatrix}, \tag{10}$$

and $\Gamma = \mathrm{diag}(\gamma_1, \ldots, \gamma_m)$. Using the inversion equations for partitioned matrices (Press et al., 1992, p. 77) we can show that

$$\Sigma_{\mathsf{xx}}^{-1} = \beta m \mathsf{I}_d - \beta \mathsf{W} \Gamma \mathsf{W}^T, \tag{11}$$

where $\Sigma_{\mathsf{xx}}$ is the covariance of the $\mathsf{x}$ variables under this model. It is easy to confirm that this is also the result obtained from summing (6) over $i = 1, \ldots, m$.

## 2.3 Maximum Likelihood solution for PoPPCA

A $m$-factor PPCA model has covariance $\sigma^2 \mathsf{I}_d + \mathsf{WW}^T$ and thus, by the Woodbury formula, it has inverse covariance $\beta \mathsf{I}_d - \beta \mathsf{W}(\sigma^2 \mathsf{I}_m + \mathsf{W}^T \mathsf{W})^{-1} \mathsf{W}^T$. The maximum likelihood solution for a $m$-PPCA model is similar to (5), i.e. $\hat{\mathsf{W}} = \mathsf{U}(\Lambda - \sigma^2 \mathsf{I}_m)^{1/2} \mathsf{R}^T$, but now $\Lambda$ is a diagonal matrix of the $m$ *principal* eigenvalues, and $\mathsf{U}$ is a matrix of the corresponding eigenvectors. If we choose $\mathsf{R}^T = \mathsf{I}$ then the columns of $\hat{\mathsf{W}}$ are orthogonal and the inverse covariance of the maximum likelihood $m$-PPCA model has the form $\beta \mathsf{I}_d - \beta \hat{\mathsf{W}} \Gamma \hat{\mathsf{W}}^T$. Comparing this to (11) (with $\mathsf{W} = \hat{\mathsf{W}}$) we see that the difference is that the first term of the RHS of (11) is $\beta m \mathsf{I}_d$, while for $m$-PPCA it is $\beta \mathsf{I}_d$.

In section 3.4 and Appendix C.3 of Agakov (2000) it is shown that (for $m \geq 2$) we obtain the $m$-factor PPCA solution when

$$\bar{\lambda} \leq \lambda_i < \frac{m}{m-1} \bar{\lambda}, \qquad i = 1, \ldots, m, \tag{12}$$

where $\bar{\lambda}$ is the mean of the $d - m$ discarded eigenvalues, and $\lambda_i$ is a retained eigenvalue; it is the smaller eigenvalues that are discarded. We see that the covariance must be nearly spherical for this condition to hold. For covariance matrices satisfying (12), this solution was confirmed by numerical experiments as detailed in (Agakov, 2000, section 3.5).

To see why this is true intuitively, observe that $\mathsf{C}_i^{-1}$ for each 1-factor PPCA expert will be large (with value $\beta$) in all directions except one. If the directions of contraction for each $\mathsf{C}_i^{-1}$ are orthogonal, we see that the sum of the inverse covariances will be at least $(m-1)\beta$ in a contracted direction and $m\beta$ in a direction in which no contraction occurs. The above shows that for certain types of sample covariance matrix the $(1, m)$ PoPPCA solution is not equivalent to the $m$-factor PPCA solution. However, it is interesting to note that by relaxing the constraint on the isotropy of each expert's noise the product of $m$ one-factor factor analysis models can be shown to be equivalent to an $m$-factor factor analyser (Marks and Movellan, 2001).

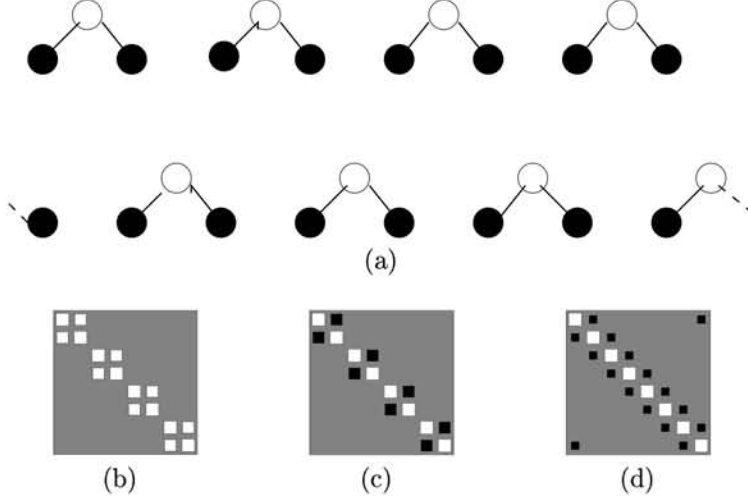

(a)

(b)        (c)        (d)

Figure 1: (a) Two experts. The upper one depicts 8 filled circles (visible units) and 4 latent variables (open circles), with connectivity as shown. The lower expert also has 8 visible and 4 latent variables, but shifted by one unit (with wraparound). (b) Covariance matrix for a single expert. (c) Inverse covariance matrix for a single expert. (d) Inverse covariance for product of experts.

## 3   A Product of Experts Representation for an AR(1) Process

For the PoPPCA case above we have considered models where the latent variables have unrestricted connectivity to the visible variables. We now consider a product of experts model with two experts as shown in Figure 1(a). The upper figure depicts 8 filled circles (visible units) and 4 latent variables (open circles), with connectivity as shown. The lower expert also has 8 visible and 4 latent variables, but shifted by one unit (with wraparound) with respect to the first expert. The 8 units are, of course, only for illustration—the construction is valid for any even number of visible units.

Consider one hidden unit and its two visible children. Denote the hidden unit by $s$ the visible units as $x_l$ and $x_r$ ($l$, $r$ for left and right). Set $s \sim N(0,1)$ and

$$x_l = as + bw_l \qquad x_r = \pm as + bw_r, \qquad (13)$$

where $w_l$ and $w_r$ are independent $N(0,1)$ random variables, and $a$, $b$ are constants. (This is a simple example of a Gaussian tree-structured process, as studied by a number of groups including that led by Prof. Willsky at MIT; see e.g. Luettgen et al. (1993).) Then $\langle x_l^2 \rangle = \langle x_r^2 \rangle = a^2 + b^2$ and $\langle x_l x_r \rangle = \pm a^2$. The corresponding $2 \times 2$ inverse covariance matrix has diagonal entries of $(a^2 + b^2)/\Delta$ and off-diagonal entries of $\mp a^2/\Delta$, where $\Delta = b^2(b^2 + 2a^2)$.

Graphically, the covariance matrix of a single expert has the form shown in Figure 1(b) (where we have used the $+$ rather than $-$ choice from (13) for all variables). Figure 1(c) shows the corresponding inverse covariance for the single expert, and Figure 1(d) shows the resulting inverse covariance for the product of the two experts, with diagonal elements $2(a^2 + b^2)/\Delta$ and off-diagonal entries of $\mp a^2/\Delta$.

An AR(1) process of the circle with $d$ nodes has the form $X_i = \alpha X_{i-1 \ (\text{mod } d)} + Z_i$,

where $Z_i \sim N(0, v)$. Thus $p(\mathsf{X}) \propto \exp -\frac{1}{2v} \sum_i (X_i - \alpha X_{i-1 \ (\text{mod } d)})^2$ and the inverse covariance matrix has a circulant tridiagonal structure with diagonal entries of $(1 + \alpha^2)/v$ and off-diagonal entries of $-\alpha/v$. The product of experts model defined above can be made equivalent to the circular AR(1) process by setting

$$ a^2 = \frac{4|\alpha|v}{(1-\alpha)^2(1+\alpha)^2}, \qquad b^2 = a^2 \frac{(1-|\alpha|)^2}{2|\alpha|}. \tag{14} $$

The $\pm$ is needed in (13) as when $\alpha$ is negative we require $x_r = -as + bw_r$ to match the inverse covariances.

We have shown that there is an exact construction to represent a stationary circular AR(1) process as a product of two Gaussian experts. The approximation of other Gaussian processes by products of tree-structured Gaussian processes is further studied in (Williams and Felderhof, 2001). Such constructions are interesting because they may allow fast approximate inference in the case that $d$ is large (and the target process may be 2 or higher dimensional) and exact inference is not tractable. Such methods have been developed by Willsky and coauthors, but not for products of Gaussians constructions.

### Acknowledgements

This work is partially supported by EPSRC grant GR/L78161 *Probabilistic Models for Sequences*. Much of the work on PoGP was carried out as part of the MSc project of FVA at the Division of Informatics, University of Edinburgh. CW thanks Sam Roweis, Geoff Hinton and Zoubin Ghahramani for helpful conversations on the *rotcaf* model during visits to the Gatsby Computational Neuroscience Unit. FVA gratefully acknowledges the support of the Royal Dutch Shell Group of Companies for his MSc studies in Edinburgh through a Centenary Scholarship. SNF gratefully acknowledges additional support from BAE Systems.

## Footnotes

[1]Because equation 3 has the form of a factor analysis decomposition, but for the *inverse* covariance matrix, we sometimes refer to PoGP as the *rotcaf* model.

### References

Agakov, F. (2000). Investigations of Gaussian Products-of-Experts Models. Master's thesis, Division of Informatics, The University of Edinburgh. Available at `http://www.dai.ed.ac.uk/homes/felixa/all.ps.gz`.

Hinton, G. E. (1999). Products of experts. In *Proceedings of the Ninth International Conference on Artificial Neural Networks (ICANN 99)*, pages 1–6.

Luettgen, M., Karl, W., and Willsky, A. (1993). Multiscale Representations of Markov Random Fields. *IEEE Trans. Signal Processing*, 41(12):3377–3395.

Marks, T. and Movellan, J. (2001). Diffusion Networks, Products of Experts, and Factor Analysis. In *Proceedings of the 3rd International Conference on Independent Component Analysis and Blind Source Separation*.

Oja, E. (1992). Principal Components, Minor Components, and Linear Neural Networks. *Neural Networks*, 5:927 – 935.

Press, W. H., Teukolsky, S. A., Vetterling, W. T., and Flannery, B. P. (1992). *Numerical Recipes in C*. Cambridge University Press, Second edition.

Proakis, J. G. and Manolakis, D. G. (1992). *Digital Signal Processing: Principles, Algorithms and Applications*. Macmillan.

Tipping, M. E. and Bishop, C. M. (1999). Probabilistic principal components analysis. *J. Roy. Statistical Society B*, 61(3):611–622.

Williams, C. K. I. and Agakov, F. V. (2001). Products of Gaussians and Probabilistic Minor Components Analysis. Technical Report EDI-INF-RR-0043, Division of Informatics, University of Edinburgh. Available at `http://www.informatics.ed.ac.uk/publications/report/0043.html`.

Williams, C. K. I. and Felderhof, S. N. (2001). Products and Sums of Tree-Structured Gaussian Processes. In *Proceedings of the ICSC Symposium on Soft Computing 2001 (SOCO 2001)*.

Xu, L. and Krzyzak, A. and Oja, E. (1991). Neural Nets for Dual Subspace Pattern Recogntion Method. *International Journal of Neural Systems*, 2(3):169–184.

# A  ML Solutions for PoGP

Here we analyze the three classes of solutions for the model covariance matrix which result from equation (4) of section 1.3.

The first case $\mathsf{W} = 0$ corresponds to a minimum of the log-likelihood.

In the second case, the model covariance $\mathsf{C}_\Sigma$ is equal to the sample covariance $\mathsf{S}$. From expression (3) for $\mathsf{C}_\Sigma^{-1}$ we find $\mathsf{W}\mathsf{W}^T = \mathsf{S}^{-1} - \beta_\Sigma \mathsf{I}_d$. This has the known solution $\mathsf{W} = \mathsf{U}_m (\Lambda^{-1} - \beta_\Sigma \mathsf{I}_m)^{1/2} \mathsf{R}^T$, where $\mathsf{U}_m$ is the matrix of the $m$ eigenvectors of $\mathsf{S}$ with the smallest eigenvalues and $\Lambda$ is the corresponding diagonal matrix of the eigenvalues. The sample covariance must be such that the largest $d - m$ eigenvalues are all equal to $\beta_\Sigma$; the other $m$ eigenvalues are matched explicitly.

Finally, for the case of approximate model covariance ($\mathsf{S}\mathsf{W} = \mathsf{C}_\Sigma\mathsf{W}$, $\mathsf{S} \neq \mathsf{C}_\Sigma$) we, by analogy with Tipping and Bishop (1999), consider the singular value decomposition of the weight matrix, and establish dependencies between left singular vectors of $\mathsf{W} = \mathsf{U}\mathsf{L}\mathsf{R}^T$ and eigenvectors of the sample covariance $\mathsf{S}$. $\mathsf{U} = [\mathsf{u}_1, \mathsf{u}_2, \ldots, \mathsf{u}_m] \subset \mathbb{R}^{d \times m}$ is a matrix of left singular vectors of $\mathsf{W}$ with columns constituting an orthonormal basis, $\mathsf{L} = \mathrm{diag}(l_1, l_2, \ldots, l_m) \subset \mathbb{R}^{m \times m}$ is a diagonal matrix of the singular values of $\mathsf{W}$ and $\mathsf{R} \subset \mathbb{R}^{m \times m}$ defines an arbitrary rigid rotation of $\mathsf{W}$. For this case equation (4) can be written as $\mathsf{S}\mathsf{U}\mathsf{L} = \mathsf{C}_\Sigma\mathsf{U}\mathsf{L}$, where $\mathsf{C}_\Sigma$ is obtained from (3) by applying the matrix inversion lemma [see e.g. Press et al. (1992)]. This leads to

$$
\begin{aligned}
\mathsf{S}\mathsf{U}\mathsf{L} = \mathsf{C}_\Sigma\mathsf{U}\mathsf{L} &= (\beta_\Sigma^{-1}\mathsf{I}_d - \beta_\Sigma^{-1}\mathsf{W}(\beta_\Sigma + \mathsf{W}^T\mathsf{W})^{-1}\mathsf{W}^T)\mathsf{U}\mathsf{L} \\
&= \mathsf{U}(\beta_\Sigma^{-1}\mathsf{I}_m - \beta_\Sigma^{-1}\mathsf{L}\mathsf{R}^T(\beta_\Sigma\mathsf{I}_m + \mathsf{R}\mathsf{L}^2\mathsf{R}^T)^{-1}\mathsf{R}\mathsf{L})\mathsf{L} \\
&= \mathsf{U}(\beta_\Sigma^{-1}\mathsf{I}_m - \beta_\Sigma^{-1}(\beta_\Sigma\mathsf{L}^{-2} + \mathsf{I}_m)^{-1})\mathsf{L}. \quad (15)
\end{aligned}
$$

Notice that the term $\beta_\Sigma^{-1}\mathsf{I}_m - \beta_\Sigma^{-1}(\beta_\Sigma\mathsf{L}^{-2} + \mathsf{I}_m)^{-1}$ in the r.h.s. of equation (15) is just a scaling factor of $\mathsf{U}$. Equation (15) defines the matrix form of the eigenvector equation, with both sides post-multiplied by the diagonal matrix $\mathsf{L}$.

If $l_i \neq 0$ then (15) implies that

$$
\mathsf{C}_\Sigma\mathsf{u}_i = \mathsf{S}\mathsf{u}_i = \lambda_i\mathsf{u}_i, \quad \lambda_i = \beta_\Sigma^{-1}(1 - (\beta_\Sigma l_i^{-2} + 1)^{-1}), \quad (16)
$$

where $\mathsf{u}_i$ is an eigenvector of $\mathsf{S}$, and $\lambda_i$ is its corresponding eigenvalue. The scaling factor $l_i$ of the $i^{th}$ retained expert can be expressed as $l_i = (\lambda_i^{-1} - \beta_\Sigma)^{1/2}$.

Obviously, if $l_i = 0$ then $\mathsf{u}_i$ is arbitrary. If $l_i = 0$ we say that the direction corresponding to $\mathsf{u}_i$ is *discarded*, i.e. the variance in that direction is explained merely by noise. Otherwise we say that $\mathsf{u}_i$ is *retained*. All potential solutions of $\mathsf{W}$ may then be expressed as

$$
\mathsf{W} = \mathsf{U}_m(\mathsf{D} - \beta_\Sigma\mathsf{I}_m)^{1/2}\mathsf{R}^T, \quad (17)
$$

where $\mathsf{R} \subset \mathbb{R}^{m \times m}$ is a rotation matrix, $\mathsf{U}_m = [\mathsf{u}_1 \mathsf{u}_2 \ldots \mathsf{u}_m] \subset \mathbb{R}^{d \times m}$ is a matrix whose columns correspond to $m$ eigenvectors of $\mathsf{S}$, and $\mathsf{D} = \mathrm{diag}(d_1, d_2, \ldots, d_m) \subset \mathbb{R}^{m \times m}$ such that $d_i = \lambda_i^{-1}$ if $\mathsf{u}_i$ is retained and $d_i = \beta_\Sigma$ if $\mathsf{u}_i$ is discarded.

It may further be shown (Williams and Agakov (2001)) that the optimal solution for the likelihood is reached when $\mathsf{W}$ corresponds to the *minor* eigenvectors of the sample covariance $\mathsf{S}$.